# Non-parametric Approximate Dynamic Programming via the Kernel Method

**Nikhil Bhat**
Graduate School of Business
Columbia University
New York, NY 10027
nbhat15@gsb.columbai.edu

**Vivek F. Farias**
Sloan School of Management
Massachusetts Institute of Technology
Cambridge, MA 02142
vivekf@mit.edu

**Ciamac C. Moallemi**
Graduate School of Business
Columbia University
New York, NY 10027
ciamac@gsb.columbai.edu

## Abstract

This paper presents a novel non-parametric approximate dynamic programming (ADP) algorithm that enjoys graceful approximation and sample complexity guarantees. In particular, we establish both theoretically and computationally that our proposal can serve as a viable alternative to state-of-the-art *parametric* ADP algorithms, freeing the designer from carefully specifying an approximation architecture. We accomplish this by developing a kernel-based mathematical program for ADP. Via a computational study on a controlled queueing network, we show that our procedure is competitive with parametric ADP approaches.

## 1 Introduction

Problems of dynamic optimization in the face of uncertainty are frequently posed as Markov decision processes (MDPs). The central computational problem is then reduced to the computation of an optimal 'cost-to-go' function that encodes the cost incurred under an optimal policy starting from any given MDP state. Many MDPs of practical interest suffer from the curse of dimensionality, where intractably large state spaces precluding exact computation of the cost-to-go function. Approximate dynamic programming (ADP) is an umbrella term for algorithms designed to produce good approximation to this function, yielding a natural 'greedy' control policy.

ADP algorithms are, in large part, *parametric* in nature; requiring the user to provide an 'approximation architecture' (i.e., a set of basis functions). The algorithm then produces an approximation in the span of this basis. The strongest theoretical results available for such algorithms typically share two features: (1) the quality of the approximation produced is comparable with the best possible within the basis specified, and (2) the computational effort required for doing so typically scales as the dimension of the basis specified.

These results highlight the importance of selecting a 'good' approximation architecture, and remain somewhat dissatisfying in that additional sampling or computational effort cannot remedy a bad approximation architecture. On the other hand, a non-parametric approach would, in principle, permit the user to select a rich, potentially full-dimensional architecture (e.g., the Haar basis). One would then expect to compute increasingly accurate approximations with increasing computational effort. The present work presents a practical algorithm of this type. Before describing our contributions, we begin with summarizing the existing body of research on non-parametric ADP algorithms.

The key computational step in approximate policy iteration methods is approximate policy evaluation. This step involves solving the projected Bellman equation, a linear stochastic fixed point equation. A numerically stable approach to this is to perform regression with a certain $\ell_2$-regularization, where the loss is the $\ell_2$-norm of the Bellman error. By substituting this step with a suitable non-parametric regression procedure, [2, 3, 4] come up with a corresponding non-parametric algorithm. Unfortunately schemes such approximate policy iteration have no convergence guarantees in parametric settings, and these difficulties remain in non-parametric variations. Another idea has been to use kernel-based local averaging ideas to approximate the solution of an MDP with that of a simpler variation on a sampled state space [5, 6, 7]. However, convergence rates for local averaging methods are exponential in the dimension of the problem state space. As in our setting, [8] constructs kernel-based cost-to-go function approximations. These are subsequently plugged into various ad hoc optimization-based ADP formulations, without theoretical justification.

Closely related to our work, [9, 10] consider modifying the approximate linear program with an $\ell_1$ regularization term to encourage sparse approximations in the span of a large, but necessarily *tractable* set of features. Along these lines, [11] discuss a non-parametric method that explicitly restricts the smoothness of the value function. However, sample complexity results for this method are not provided and it appears unsuitable for high-dimensional problems (such as, for instance, the problem we consider in our experiments). In contrast to this line of work, our approach will allow for approximations in a potentially infinite dimensional approximation architecture with a constraint on an appropriate $\ell_2$-norm of the weight vector.

The non-parametric ADP algorithm we develop enjoys non-trivial approximation and sample complexity guarantees. We show that our approach complements state-of-the-art parametric ADP algorithms by allowing the algorithm designer to compute what is essentially the best possible 'simple' approximation[1] in a full-dimensional approximation architecture as opposed to restricting attention to some a-priori fixed low dimensional architecture. In greater detail, we make the following contributions:

*A new mathematical programming formulation.* We rigorously develop a kernel-based variation of the 'smoothed' approximate LP (SALP) approach to ADP proposed by [12]. The resulting mathematical program, which we dub the regularized smoothed approximate LP (RSALP), is distinct from simply substituting a kernel-based approximation in the SALP formulation. We develop a companion active set method that is capable of solving this mathematical program rapidly and with limited memory requirements.

*Theoretical guarantees.* [2] We establish a graceful approximation guarantee for our algorithm. Our algorithm can be interpreted as solving an approximate linear program in an appropriate Hilbert space. We provide, with high probability, an upper bound on the approximation error of the algorithm relative to the best possible approximation subject to a regularization constraint. The sampling requirements for our method are, in fact, *independent* of the dimension of the approximation architecture. Instead, we show that the number of samples grows polynomially as a function of a regularization parameter. Hence, the sampling requirements are a function of the complexity of the approximation, not of the dimension of the approximating architecture. This result can be seen as the 'right' generalization of the prior parametric approximate LP approaches [13, 14, 12], where, in contrast, sample complexity grows with the dimension of the approximating architecture.

*A computational study.* To study the efficacy of RSALP, we consider an MDP arising from a challenging queueing network scheduling problem. We demonstrate that our RSALP method yields significant improvements over known heuristics and standard parametric ADP methods.

In what follows, proofs and a detailed discussion of our numerical procedure are deferred to the Online Supplement to this paper.

## 2  Formulation

Consider a discrete time Markov decision process with finite state space $\mathcal{S}$ and finite action space $\mathcal{A}$. We denote by $x_t$ and $a_t$ respectively, the state and action at time $t$. We assume time-homogeneous Markovian dynamics: conditioned on being at state $x$ and taking action $a$, the system transitions to state $x'$ with probability $p(x, x', a)$ independent of the past. A policy is a map $\mu\colon \mathcal{S} \to \mathcal{A}$, so that

$$J^\mu(x) \triangleq \mathsf{E}_{x,\mu}\left[\sum_{t=0}^{\infty} \alpha^t g_{x_t,a_t}\right]$$

represents the expected (discounted, infinite horizon) cost-to-go under policy $\mu$ starting at state $x$. Letting $\Pi$ denote the set of all policies our goal is to find an optimal policy $\mu^*$ such that $\mu^* \in \operatorname{argmax}_{\mu \in \Pi} J^\mu(x)$ for all $x \in \mathcal{S}$ (it is well known that such a policy exists). We denote the optimal cost-to-go function by $J^* \triangleq J^{\mu^*}$. An optimal policy $\mu^*$ can be recovered as a 'greedy' policy with respect to $J^*$,

$$\mu^*(x) \in \operatorname*{argmin}_{a \in \mathcal{A}}\ g_{x,a} + \alpha \mathsf{E}_{x,a}[J^*(X')],$$

where we define $\mathsf{E}_{x,a}[f(X')]$ as $\sum_{x' \in \mathcal{S}} p(x, x', a) f(x')$, for all $f\colon \mathcal{S} \to \mathbb{R}$.

Since in practical applications $\mathcal{S}$ is often intractably large, exact computation of $J^*$ is untenable. ADP algorithms are principally tasked with computing approximations to $J^*$ of the form $J^*(x) \approx z^\top \Phi(x) \triangleq \tilde{J}(x)$, where $\Phi\colon \mathcal{S} \to \mathbb{R}^m$ is referred to as an 'approximation architecture' or a basis and must be provided as input to the ADP algorithm. The ADP algorithm computes a 'weight' vector $z$; one then employs a policy that is greedy with respect to the corresponding approximation $\tilde{J}$.

### 2.1  Primal Formulation

Motivated by the LP for exact dynamic programming, a series of ADP algorithms [15, 13, 12] have been proposed that compute a weight vector $z$ by solving an appropriate modification of the exact LP for dynamic programming. In particular, [12] propose solving the following optimization problem where $\nu \in \mathbb{R}_+^\mathcal{S}$ is a strictly positive probability distribution and $\kappa > 0$ is a penalty parameter:

$$
\begin{aligned}
\max \quad & \sum_{x \in \mathcal{S}} \nu_x z^\top \Phi(x) - \kappa \sum_{x \in \mathcal{S}} \pi_x s_x \\
\text{s. t.} \quad & z^\top \Phi(x) \leq g_{a,x} + \alpha \mathsf{E}_{x,a}[z^\top \Phi(X')] + s_x, \quad \forall\, x \in \mathcal{S},\ a \in \mathcal{A}, \\
& z \in \mathbb{R}^m, s \in \mathbb{R}_+^\mathcal{S}.
\end{aligned}
\tag{1}
$$

In parsing the above program notice that if one insisted that the slack variables $s$ were precisely $0$, one is left with the ALP proposed by [15]. [13] provided a pioneering analysis that loosely showed

$$\|J^* - z^{*\top}\Phi\|_{1,\nu} \leq \frac{2}{1-\alpha} \inf_z\ \|J^* - z^\top \Phi\|_\infty,$$

for an optimal solution $z^*$ to the ALP; [12] showed that these bounds could be improved upon substantially by 'smoothing' the constraints of the ALP, i.e., permitting positive slacks. In both cases, one must solve a 'sampled' version of the above program.

Now, consider allowing $\Phi$ to map from $\mathcal{S}$ to a general (potentially infinite dimensional) Hilbert space $\mathcal{H}$. We use bold letters to denote elements in the Hilbert space $\mathcal{H}$, e.g., the weight vector is denoted by $\mathbf{z} \in \mathcal{H}$. We further suppress the dependence on $\Phi$ and denote the elements $\mathcal{H}$ corresponding to their counterparts in $\mathcal{S}$ by bold letters. Hence, for example, $\mathbf{x} \triangleq \Phi(x)$ and $\mathbf{X} \triangleq \Phi(X)$. Further, we denote $\mathcal{X} \triangleq \Phi(\mathcal{S})$; $\mathcal{X} \subset \mathcal{H}$. The value function approximation in this case would be given by

$$\tilde{J}_{\mathbf{z},b}(x) \triangleq \langle \mathbf{x}, \mathbf{z}\rangle + b = \langle \Phi(x), \mathbf{z}\rangle + b, \tag{2}$$

where $b$ is a scalar offset corresponding to a constant basis function. The following generalization of (1) — which we dub the regularized SALP (RSALP) — then essentially suggests itself:

$$
\begin{aligned}
\max \quad & \sum_{x \in \mathcal{S}} \nu_x \langle \mathbf{x}, \mathbf{z}\rangle + b - \kappa \sum_{x \in \mathcal{S}} \pi_x s_x - \frac{\Gamma}{2}\langle \mathbf{z}, \mathbf{z}\rangle \\
\text{s. t.} \quad & \langle \mathbf{x}, \mathbf{z}\rangle + b \leq g_{a,x} + \alpha \mathsf{E}_{x,a}[\langle \mathbf{X}', \mathbf{z}\rangle + b] + s_x, \quad \forall\, x \in \mathcal{S},\ a \in \mathcal{A}, \\
& \mathbf{z} \in \mathcal{H},\ b \in \mathbb{R},\ s \in \mathbb{R}_+^\mathcal{S}.
\end{aligned}
\tag{3}
$$

The only 'new' ingredient in the program above is the fact that we regularize $\mathbf{z}$ using the parameter $\Gamma > 0$. Constraining $\|\mathbf{z}\|_{\mathcal{H}} \triangleq \sqrt{\langle z, z \rangle}$ to lie within some $\ell_2$-ball anticipates that we will eventually resort to sampling in solving this program and we cannot hope for a reasonable number of samples to provide a good solution to a problem where $\mathbf{z}$ was unconstrained. This regularization, which plays a crucial role both in theory and practice, is easily missed if one directly 'plugs in' a local averaging approximation in place of $z^\top \Phi(x)$ as is the case in the earlier work of [5, 6, 7, 8] and others.

Since the RSALP, i.e., program (3), can be interpreted as a regularized stochastic optimization problem, one may hope to solve it via its sample average approximation. To this end, define the likelihood ratio $w_x \triangleq \nu_x/\pi_x$, and let $\hat{\mathcal{S}} \subset \mathcal{S}$ be a set of $N$ states sampled independently according to the distribution $\pi$. The sample average approximation of (3) is then

$$
\begin{aligned}
\max \quad & \frac{1}{N}\sum_{x\in\hat{\mathcal{S}}} w_x \langle \mathbf{x}, \mathbf{z}\rangle + b - \frac{\kappa}{N}\sum_{x\in\hat{\mathcal{S}}} s_x - \frac{\Gamma}{2}\langle \mathbf{z}, \mathbf{z}\rangle \\
\text{s. t.} \quad & \langle \mathbf{x}, \mathbf{z}\rangle + b \le g_{a,x} + \alpha \mathsf{E}_{x,a}[\langle \mathbf{X}', \mathbf{z}\rangle + b] + s_x, \quad \forall\, x\in\hat{\mathcal{S}},\; a\in\mathcal{A}, \\
& \mathbf{z}\in\mathcal{H},\; b\in\mathbb{R},\; s\in\mathbb{R}_+^{\hat{\mathcal{S}}}.
\end{aligned}
\tag{4}
$$

We call this program the sampled RSALP. Even if $|\hat{\mathcal{S}}|$ were small, it is still not clear that this program can be solved effectively. We will, in fact, solve the dual to this problem.

## 2.2 Dual Formulation

We begin by establishing some notation. Let $\mathcal{N}_{x,a} \triangleq \{x\} \cup \{x' \in \mathcal{S}\,|\,p(x, x', a) > 0\}$. Now, define the symmetric positive semi-definite matrix $Q \in \mathbb{R}^{(\hat{\mathcal{S}}\times\mathcal{A})\times(\hat{\mathcal{S}}\times\mathcal{A})}$ according to

$$
Q(x, a, x', a') \triangleq \sum_{y\in\mathcal{N}_{x,a}} \sum_{y'\in\mathcal{N}_{x',a'}} \left(\mathbf{1}_{\{x=y\}} - \alpha p(x, y, a)\right)\left(\mathbf{1}_{\{x'=y'\}} - \alpha p(x', y', a)\right)\langle \mathbf{y}, \mathbf{y}'\rangle,
\tag{5}
$$

and the vector $R \in \mathbb{R}^{\hat{\mathcal{S}}\times\mathcal{A}}$ according to

$$
R(x, a) \triangleq \Gamma g_{x,a} - \frac{1}{N}\sum_{x'\in\hat{\mathcal{S}}}\sum_{y\in\mathcal{N}_{x,a}} w_{x'}\left(\mathbf{1}_{\{x=y\}} - \alpha p(x, y, a)\right)\langle \mathbf{y}, \mathbf{x}'\rangle.
\tag{6}
$$

Notice that $Q$ and $R$ depend only on inner products in $\mathcal{X}$ (and other, easily computable quantities). The dual to (4) is then given by:

$$
\begin{aligned}
\min \quad & \tfrac{1}{2}\lambda^\top Q \lambda + R^\top \lambda \\
\text{s. t.} \quad & \sum_{a\in\mathcal{A}} \lambda_{x,a} \le \frac{\kappa}{N}, \qquad \forall\; x\in\hat{\mathcal{S}}, \\
& \sum_{x\in\hat{\mathcal{S}}}\sum_{a\in\mathcal{A}} \lambda_{x,a} = \frac{1}{1-\alpha}, \quad \lambda\in\mathbb{R}_+^{\hat{\mathcal{S}}\times\mathcal{A}}.
\end{aligned}
\tag{7}
$$

Assuming that $Q$ and $R$ can be easily computed, this finite dimensional quadratic program, *is* tractable – its size is polynomial in the number of sampled states. We may recover a primal solution (i.e., the weight vector $\mathbf{z}^*$) from an optimal dual solution:

**Proposition 1.** *The optimal solution to* (7) *is attained at some $\lambda^*$, then optimal solution to* (4) *is attained at some $(z^*, s^*, b^*)$ with*

$$
\mathbf{z}^* = \frac{1}{\Gamma}\left[\frac{1}{N}\sum_{x\in\hat{\mathcal{S}}} w_x \mathbf{x} - \sum_{x\in\hat{\mathcal{S}}, a\in\mathcal{A}} \lambda_{x,a}^*\left(\mathbf{x} - \alpha\mathsf{E}_{x,a}[\mathbf{X}']\right)\right].
\tag{8}
$$

Having solved this program, we may, using Proposition 1, recover our approximate cost-to-go function $\tilde{J}(x) = \langle \mathbf{z}^*, \mathbf{x}\rangle + b^*$ as

$$
\tilde{J}(x) = \frac{1}{\Gamma}\left[\frac{1}{N}\sum_{y\in\hat{\mathcal{S}}} w_y \langle \mathbf{y}, \mathbf{x}\rangle - \sum_{y\in\hat{\mathcal{S}}, a\in\mathcal{A}} \lambda_{y,a}^*\left(\langle \mathbf{y}, \mathbf{x}\rangle - \alpha\mathsf{E}_{y,a}[\langle \mathbf{X}', \mathbf{x}\rangle]\right)\right] + b^*.
\tag{9}
$$

A policy greedy with respect to $\tilde{J}$ is not affected by constant translations, hence in (9), the value of $b^*$ can be set to be zero arbitrarily. Again note that given $\lambda^*$, $\tilde{J}$ only involves the inner products.

At this point, we use the 'kernel' trick: instead of explicitly specifying $\mathcal{H}$ or the mapping $\Phi$, we take the approach of specifying inner products. In particular, given any positive definite kernel $K\colon \mathcal{S} \times \mathcal{S} \to \mathbb{R}$, it is well known (Mercer's theorem) that there exists a Hilbert space $\mathcal{H}$ and $\Phi\colon \mathcal{S} \to \mathcal{H}$ such that $K(x,y) = \langle \Phi(x), \Phi(y) \rangle$. Consequently, given a positive definite kernel, we simply replace every inner product $\langle \mathbf{x}, \mathbf{x}' \rangle$ in the defining of the program (7) with the quantity $K(x, x')$ and similarly in the approximation (9). In particular, this is equivalent to using a Hilbert space, $\mathcal{H}$ and mapping $\Phi$ corresponding to that kernel.

Solving (7) directly is costly. In particular, it is computationally expensive to pre-compute and store the matrix $Q$. An alternative to this is to employ the following broad strategy, as recognized by [16] and [17] in the context of solving SVM classification problems, referred to as an active set method: At every point in time, one attempts to (a) change only a small number of variables while not impacting other variables (b) maintain feasibility. It turns out that this results in a method that requires memory and per-step computation that scales only linearly with the sample size. We defer the details of the procedure as well as the theoretical analysis to the Online Supplement

# 3 Approximation Guarantees

Recall that we are employing an approximation $\tilde{J}_{\mathbf{z},b}$ of the form (2), parameterized by the weight vector $\mathbf{z}$ and the offset parameter $b$. Now denoting by $\mathcal{C}$ the feasible region of the RSALP projected onto the $\mathbf{z}$ and $b$ co-ordinates, the *best* possible approximation one may hope for among those permitted by the RSALP will have $\ell_\infty$-approximation error $\inf_{(\mathbf{z},b)\in\mathcal{C}} \|J^* - \tilde{J}_{\mathbf{z},b}\|_\infty$. Provided the Gram matrix given by the kernel restricted to $\mathcal{S}$ is positive definite, this quantity can be made arbitrarily small by making $\Gamma$ small. The rate at which this happens would reflect the quality of the kernel in use. Here we focus on asking the following question: for a fixed choice of regularization parameters (i.e., with $\mathcal{C}$ fixed) what approximation guarantee can be obtained for a solution to the RSALP? This section will show that one can achieve a guarantee that is, in essence, within a certain constant multiple of the optimal approximation error using a number of samples that is independent of the size of the state space and the dimension of the approximation architecture.

## 3.1 The Guarantee

Define the Bellman operator, $T\colon \mathbb{R}^\mathcal{S} \to \mathbb{R}^\mathcal{S}$ according to

$$(TJ)(x) \triangleq \min_{a\in\mathcal{A}} \ g_{x,a} + \alpha \mathsf{E}_{x,a}[J(X')].$$

Let $\hat{\mathcal{S}}$ be a set of $N$ states drawn independently at random from $\mathcal{S}$ under the distribution $\pi$ over $\mathcal{S}$. Given the definition of $\tilde{J}_{\mathbf{z},b}$ in (2), we consider the following sampled version of RSALP,

$$
\begin{aligned}
\max \quad & \nu^\top \tilde{J}_{\mathbf{z},b} - \frac{2}{1-\alpha}\frac{1}{N}\sum_{x\in\hat{\mathcal{S}}} s_x \\
\text{s.\,t.} \quad & \langle \mathbf{x}, \mathbf{z} \rangle + b \leq g_{a,x} + \alpha \mathsf{E}_{x,a}[\langle \mathbf{X}', \mathbf{z}\rangle + b] + s_x, \quad \forall\, x \in \hat{\mathcal{S}},\ a \in \mathcal{A}, \\
& \|\mathbf{z}\|_\mathcal{H} \leq C, \ |b| \leq B, \qquad\qquad\qquad\qquad \mathbf{z}\in\mathcal{H},\ b\in\mathbb{R},\ s\in\mathbb{R}_+^{\hat{\mathcal{S}}}.
\end{aligned}
\tag{10}
$$

We will assume that states are sampled according to an idealized distribution. In particular, $\pi \triangleq \pi_{\mu^*,\nu}$ where

$$\pi_{\mu^*,\nu}^\top \triangleq (1-\alpha)\sum_{t=0}^\infty \alpha^t \nu^\top P_{\mu^*}^t. \tag{11}$$

Here, $P_{\mu^*}$ is the transition matrix under the optimal policy $\mu^*$. This idealized assumption is also common to the work of [14] and [12]. In addition, this program is somewhat distinct from the program presented earlier, (4): (1) As opposed to a 'soft' regularization term in the objective, we have a 'hard' regularization constraint, $\|\mathbf{z}\|_\mathcal{H} \leq C$. It is easy to see that given a $\Gamma$, we can choose a radius $C(\Gamma)$ that yields an equivalent optimization problem. (2) We bound the magnitude of the offset $b$. This is for theoretical convenience; our sample complexity bound will be parameterized

by $B$. (3) We fix $\kappa = 2/(1 - \alpha)$. Our analysis reveals this to be the 'right' penalty weight on the Bellman inequality violations.

Before stating our bound we establish a few bits of notation. We let $(\mathbf{z}^*, b^*)$ denote an optimal solution to (10). We let $K \triangleq \max_{x \in \mathcal{X}} \|\mathbf{x}\|_{\mathcal{H}}$, and finally, we define the quantity

$$\Xi(C, B, K, \delta) \quad \triangleq \quad \left(1 + \sqrt{\frac{1}{2}\ln(1/\delta)}\right)\left(4CK(1 + \alpha) + 4B(1 - \alpha) + 2\|g\|_\infty\right).$$

We have the following theorem:

**Theorem 1.** *For any $\epsilon > 0$ and $\delta > 0$, let $N \geq \Xi(C, B, K, \delta)^2/\epsilon^2$. If (10) is solved by sampling $N$ states from $\mathcal{S}$ with distribution $\pi_{\mu^*, \nu}$, then with probability at least $1 - \delta - \delta^4$,*

$$\|J^* - \tilde{J}_{\mathbf{z}^*, b^*}\|_{1, \nu} \leq \inf_{\|\mathbf{z}\|_{\mathcal{H}} \leq C, |b| \leq B} \frac{3 + \alpha}{1 - \alpha}\|J^* - \tilde{J}_{\mathbf{z}, b}\|_\infty + \frac{4\epsilon}{1 - \alpha}. \tag{12}$$

Ignoring the $\epsilon$-dependent error terms, we see that the quality of approximation provided by $(\mathbf{z}^*, b^*)$ is essentially within a constant multiple of the optimal (in the sense of $\ell_\infty$-error) approximation to $J^*$ possible using a weight vector $\mathbf{z}$ and offsets $b$ permitted by the regularization constraints. This is a 'structural' error term that will persist even if one were permitted to draw an arbitrarily large number of samples. It is analogous to the approximation results produced in parametric settings with the important distinction that *one allows comparisons to approximations in potentially full-dimensional basis sets* which might be substantially superior.

In addition to the structural error above, one incurs an additional additive 'sampling' error that scales like $O(N^{-1/2}(CK + B)\sqrt{\ln 1/\delta})$. This quantity has no explicit dependence on the dimension of the approximation architecture. In contrast, comparable sample complexity results (eg. [14, 12]) typically scale with the dimension of the approximation architecture. Here, this space may be full dimensional, so that such a dependence would yield a vacuous guarantee. The error depends on the user specified quantities $C$ and $B$, and $K$, which is bounded for many kernels. The result allows for arbitrary 'simple' (i.e. with $\|\mathbf{z}\|_{\mathcal{H}}$ small) approximations in a rich feature space as opposed to restricting us to some a-priori fixed, low dimensional feature space. This yields some intuition for why we expect the approach to perform well even with a relatively general choice of kernel.

As $C$ and $B$ grow large, the structural error will decrease to zero provided $K$ restricted to $\mathcal{S}$ is positive definite. In order to maintain the sampling error constant, one would then need to increase $N$ (at a rate that is $\Omega((CK + B)^2)$). In summary, increased sampling yields approximations of increasing quality, approaching an *exact* approximation. If $J^*$ admits a good approximation with $\|\mathbf{z}\|_{\mathcal{H}}$ small, one can expect a good approximation with a reasonable number of samples.

### 3.2 Proof Sketch

A detailed proof of a stronger result is in the Online Supplement. Here, we provide a proof sketch.

The first step of the proof involves providing a guarantee for the exact (non-sampled) RSALP with hard regularization. Assuming $(\mathbf{z}^*, b^*)$ is the 'learned' parameter pair, we first establish the guarantee:

$$\|J^* - \tilde{J}_{\mathbf{z}^*, b^*}\|_{1, \nu} \leq \frac{3 + \alpha}{1 - \alpha} \inf_{\|\mathbf{z}\|_{\mathcal{H}} \leq C, b \in \mathbb{R}} \|J^* - \tilde{J}_{\mathbf{z}, b}\|_\infty.$$

Geometrically, the proof works loosely by translating the 'best' approximation given the regularization constraints to one that is guaranteed to yield an approximation error no worse that that produced by the RSALP.

To establish a guarantee for the *sampled* RSALP, we first pose the RSALP as a stochastic optimization problem by setting $s(\mathbf{z}, b) \triangleq (\tilde{J}_{\mathbf{z}, b} - T\tilde{J}_{\mathbf{z}, b})^+$. We must ensure that with high probability, the sample averages in the sampled program are close to the exact expectations, *uniformly* for all possible values of $(\mathbf{z}, b)$ with high accuracy. In order to establish such a guarantee, we bound the Rademacher complexity of the class of functions given by

$$\bar{\mathcal{F}}_{\mathcal{S}, \mu} \triangleq \left\{x \mapsto (\tilde{J}_{\mathbf{z}, b}(x) - T_\mu \tilde{J}_{\mathbf{z}, b}(x))^+ \; : \; \|z\|_{\mathcal{H}} \leq C, |b| \leq B\right\},$$

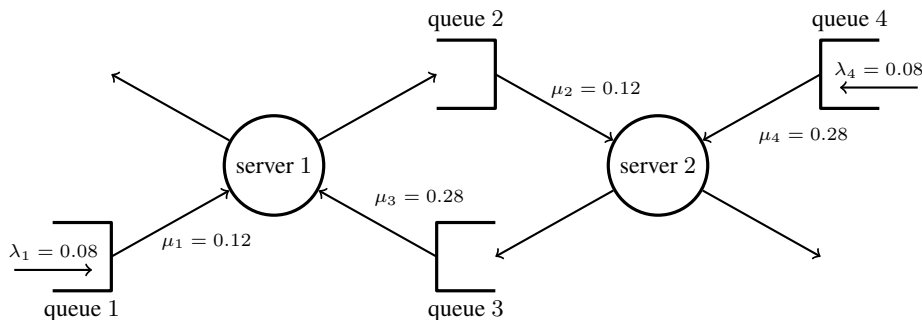

Figure 1: The queueing network example.

(where $T_\mu$ is the Bellman operator associated with policy $\mu$), This yields the appropriate uniform large deviations bound. Using this guarantee we show that the optimal solution to the sampled RSALP yields similar approximation guarantees as that with the exact RSALP; this proof is somewhat delicate as it appears difficult to directly show that the optimal solutions themselves are close.

## 4 Case Study: A Queueing Network

This section considers the problem of controlling the queuing network illustrated in Figure 1, with the objective of minimizing long run average delay. There are two 'flows' in this network: the first through server 1 followed by server 2 (with buffering at queues 1 and 2, respectively), and the second through server 2 followed by server 1 (with buffering at queues 4 and 3, respectively). Here, all inter-arrival and service times are exponential with rate parameters summarized in Figure 1. This specific network has been studied [13, 18] and is considered to be a challenging control problem. Our goal in this section will be two-fold. First, we will show that the RSALP can surpass the performance of both heuristic as well as established ADP-based approaches, when used 'out-of-the-box' with a generic kernel. Second, we will show that the RSALP can be solved efficiently.

### 4.1 MDP Formulation

Although the control problem at hand is nominally a continuous time problem, it is routinely converted into a discrete time problem via a standard uniformization device; see [19], for instance, for an explicit such example. In the equivalent discrete time problem, at most a single event can occur in a given epoch, corresponding either to the arrival of a job at queues 1 or 4, or the arrival of a service token for one of the four queues with probability proportional to the corresponding rates. The state of the system is described by the number of jobs is each of the four queues, so that $\mathcal{S} \triangleq \mathbb{Z}_+^4$, whereas the action space $\mathcal{A}$ consists of four potential actions each corresponding to a matching between servers and queues. We take the single period cost as the total number of jobs in the system, so that $g_{x,a} = \|x\|_1$; note that minimizing the average number of jobs in the system is equivalent to minimizing average delay by Little's law. Finally, we take $\alpha = 0.9$ as our discount factor.

### 4.2 Approaches

**RSALP (this paper).** We solve (7) using the active set method outlined in the Online Supplement, taking as our kernel the standard Gaussian radial basis function kernel $K(x,y) \triangleq \exp\left(-\|x-y\|_2^2/h\right)$, with the bandwidth parameter $h \triangleq 100$. (The sensitivity of our results to this bandwidth parameter appears minimal.) Note that this implicitly corresponds to a full-dimensional basis function architecture. Since the idealized sampling distribution, $\pi_{\mu^*,\nu}$ is unavailable to us, we use in its place the geometric distribution $\pi(x) \triangleq (1-\zeta)^4 \zeta^{\|x\|_1}$, with the sampling parameter $\zeta$ set at 0.9, as in [13]. The regularization parameter $\Gamma$ was chosen via a line-search; we report results for $\Gamma \triangleq 10^{-8}$. (Again performance does not appear to be very sensitive to $\Gamma$, so that a crude line-search appears to suffice.) In accordance with the theory we set the constraint violation parameter $\kappa \triangleq 2/(1-\alpha)$, as suggested by the analysis of Section 3.1, as well as by [12],

| policy | performance | | | | | | |
|---|---|---|---|---|---|---|---|
| Longest Queue | 8.09 | | | | | | |
| Max-Weight | 6.55 | | | | | | |
| **sample size** | 1000 | | 3000 | | 5000 | | 10000 |
| SALP, cubic basis | 7.19 | (1.76) | 7.89 | (1.76) | 6.94 | (1.15) | 6.63 (0.92) |
| RSALP, Gaussian kernel | 6.72 | (0.39) | 6.31 | (0.11) | 6.13 | (0.08) | 6.04 (0.05) |

Table 1: Performance results in the queueing example. For the SALP and RSALP methods, the number in the parenthesis gives the standard deviation across sample sets.

**SALP [12].** The SALP formulation (1), is, as discussed earlier, the parametric counterpart to the RSALP. It may be viewed as a generalization of the ALP approach proposed by [13] and has been demonstrated to provide substantial performance benefits relative to the ALP approach. Our choice of parameters for the SALP mirrors those for the RSALP to the extent possible, so as to allow for an 'apples-to-apples' comparison. Thus, we solve the sample average approximation of this program using the same geometric sampling distribution and parameter $\kappa$. Approximation architectures in which the basis functions are monomials of the queue lengths appear to be a popular choice for queueing control problems [13]. We use all monomials with degree at most 3, which we will call the *cubic basis*, as our approximation architectures.

**Longest Queue (generic).** This is a simple heuristic approach: at any given time, a server chooses to work on the longest queue from among those it can service.

**Max-Weight [20].** Max-Weight is a well known scheduling heuristic for queueing networks. The policy is obtained as the greedy policy with respect to a value function approximation of the form $\tilde{J}_{MW}(x) \triangleq \sum_{i=1}^{4} |x_i|^{1+\epsilon}$, given a parameter $\epsilon > 0$. This policy has been extensively studied and shown to have a number of good properties, for example, being throughput optimal and offering good performance for critically loaded settings [21]. Via a line-search, we chose to $\epsilon \triangleq 1.5$ as the exponent for our experiments.

### 4.3 Results

Policies were evaluated using a common set of arrival process sample paths. The performance metric we report for each control policy is the long run average number of jobs in the system under that policy, $\sum_{t=1}^{T} \|x_t\|_1/T$, where we set $T \triangleq 10000$. We further average this random quantity over an ensemble of 300 sample paths. Further, in order to generate SALP and RSALP policies, state sampling is required. To understand the effect of the sample size on the resulting policy performance, the different sample sizes listed in Table 1 were used. Since the policies generated involve randomness to the sampled states, we further average performance over 10 sets of sampled states. The results are reported in Table 1 and have the following salient features:

1. *RSALP outperforms established policies:* Approaches such as the Max-Weight or 'parametric' ADP with basis spanning polynomials have been previously shown to work well for the problem of interest. We see that RSALP with 10000 samples achieves performance that is superior to these extant schemes.

2. *Sampling improves performance:* This is expected from the theory in Section 3. Ideally, as the sample size is increased one should relax the regularization. However, for our experiments we noticed that the performance is quite insensitive to the parameter $\Gamma$. Nonetheless, it is clear that larger sample sets yield a significant performance improvement.

3. *RSALP in less sensitive to state sampling:* We notice from the standard deviation values in Table 1 that our approach gives policies whose performance varies significantly less across different sample sets of the same size.

In summary we view these results as indicative of the possibility that the RSALP may serve as a practical and viable alternative to state-of-the-art parametric ADP techniques.

## Footnotes

[1]In the sense that the $\ell_2$ norm of the weight vector can grow at most polynomially with a certain measure of computational budget.

[2]These guarantees come under assumption of being able to sample from a certain idealized distribution. This is a common in the ADP literature.

# References

[1] D. P. Bertsekas. *Dynamic Programming and Optimal Control, Vol. II*. Athena Scientific, 2007.

[2] B. Bethke, J. P. How, and A. Ozdaglar. Kernel-based reinforcement learning using Bellman residual elimination. MIT Working Paper, 2008.

[3] Y. Engel, S. Mannor, and R. Meir. Bayes meets Bellman: The Gaussian process approach to temporal difference learning. In *Proceedings of the 20th International Conference on Machine Learning*, pages 154–161. AAAI Press, 2003.

[4] X. Xu, D. Hu, and X. Lu. Kernel-based least squares policy iteration for reinforcement learning. *IEEE Transactions on Neural Networks*, 18(4):973–992, 2007.

[5] D. Ormoneit and S. Sen. Kernel-based reinforcement learning. *Machine Learning*, 49(2):161–178, 2002.

[6] D. Ormoneit and P. Glynn. Kernel-based reinforcement learning in average cost poblems. *IEEE Transactions on Automatic Control*, 47(10):1624–1636, 2002.

[7] A. M. S. Barreto, D. Precup, and J. Pineau. Reinforcement learning using kernel-based stochastic factorization. In *Advances in Neural Information Processing Systems*, volume 24, pages 720–728. MIT Press, 2011.

[8] T. G. Dietterich and X. Wang. Batch value function approximation via support vectors. In *Advances in Neural Information Processing Systems*, volume 14, pages 1491–1498. MIT Press, 2002.

[9] J. Kolter and A. Ng. Regularization and feature selection in least-squares temporal difference learning. ICML '09, pages 521–528. ACM, 2009.

[10] M. Petrik, G. Taylor, R. Parr, and S. Zilberstein. Feature selection using regularization in approximate linear programs for Markov decision processes. ICML '10, pages 871–879, 2010.

[11] J. Pazis and R. Parr. Non-parametric approximate linear programming for MDPs. AAAI Conference on Artificial Intelligence. AAAI, 2011.

[12] V. V. Desai, V. F. Farias, and C. C. Moallemi. Approximate dynamic programming via a smoothed linear program. To appear in *Operations Research*, 2011.

[13] D. P. de Farias and B. Van Roy. The linear programming approach to approximate dynamic programming. *Operations Research*, 51(6):850–865, 2003.

[14] D. P. de Farias and B. Van Roy. On constraint sampling in the linear programming approach to approximate dynamic programming. *Mathematics of Operations Research*, 29:3:462–478, 2004.

[15] P. Schweitzer and A. Seidman. Generalized polynomial approximations in Markovian decision processes. *Journal of Mathematical Analysis and Applications*, 110:568–582, 1985.

[16] E. Osuna, R. Freund, and F. Girosi. An improved training algorithm for support vector machines. In *Neural Networks for Signal Processing, Proceedings of the 1997 IEEE Workshop*, pages 276 –285, sep 1997.

[17] T. Joachims. *Making large-scale support vector machine learning practical*, pages 169–184. MIT Press, Cambridge, MA, USA, 1999.

[18] R. R. Chen and S. Meyn. Value iteration and optimization of multiclass queueing networks. In *Decision and Control, 1998. Proceedings of the 37th IEEE Conference on*, volume 1, pages 50 –55 vol.1, 1998.

[19] C. C. Moallemi, S. Kumar, and B. Van Roy. Approximate and data-driven dynamic programming for queueing networks. Working Paper, 2008.

[20] L. Tassiulas and A. Ephremides. Stability properties of constrained queueing systems and scheduling policies for maximum throughput in multihop radio networks. *IEEE Transactions on Automatic Control*, 37(12):1936–1948, December 1992.

[21] A. L. Stolyar. Maxweight scheduling in a generalized switch: State space collapse and workload minimization in heavy traffic. *The Annals of Applied Probability*, 14:1–53, 2004.

